# The Parallel Problems Server: an Interactive Tool for Large Scale Machine Learning

**Charles Lee Isbell, Jr.**
isbell@research.att.com
AT&T Labs
180 Park Avenue Room A255
Florham Park, NJ 07932-0971

**Parry Husbands**
PJRHusbands@lbl.gov
Lawrence Berkeley National Laboratory/NERSC
1 Cyclotron Road, MS 50F
Berkeley, CA 94720

## Abstract

Imagine that you wish to classify data consisting of tens of thousands of examples residing in a twenty thousand dimensional space. How can one apply standard machine learning algorithms? We describe the Parallel Problems Server (PPServer) and MATLAB*P. In tandem they allow users of networked computers to work transparently on large data sets from within Matlab. This work is motivated by the desire to bring the many benefits of scientific computing algorithms and computational power to machine learning researchers.

We demonstrate the usefulness of the system on a number of tasks. For example, we perform *independent components analysis* on very large text corpora consisting of tens of thousands of documents, making minimal changes to the original Bell and Sejnowski Matlab source (Bell and Sejnowski, 1995). Applying ML techniques to data previously beyond their reach leads to interesting analyses of both data and algorithms.

## 1 Introduction

Real-world data sets are extremely large by the standards of the machine learning community. In text retrieval, for example, we often wish to process collections consisting of tens or hundreds of thousands of documents and easily as many different words. Naturally, we would like to apply machine learning techniques to this problem; however, the sheer size of the data makes this difficult.

This paper describes the Parallel Problems Server (PPServer) and MATLAB*P. The PPServer is a "linear algebra server" that executes distributed memory algorithms on large data sets. Together with MATLAB*P, users can manipulate large data sets within Matlab transparently. This system brings the efficiency and power of highly-optimized parallel computation to researchers using networked machines but maintain the many benefits of interactive environments.

We demonstrate the usefulness of the PPServer on a number of tasks. For example, we perform *independent components analysis* on very large text corpora consisting of tens of thousands of documents with minimal changes to the original Bell and Sejnowski Matlab source (Bell and Sejnowski, 1995). Applying ML techniques to datasets previously beyond

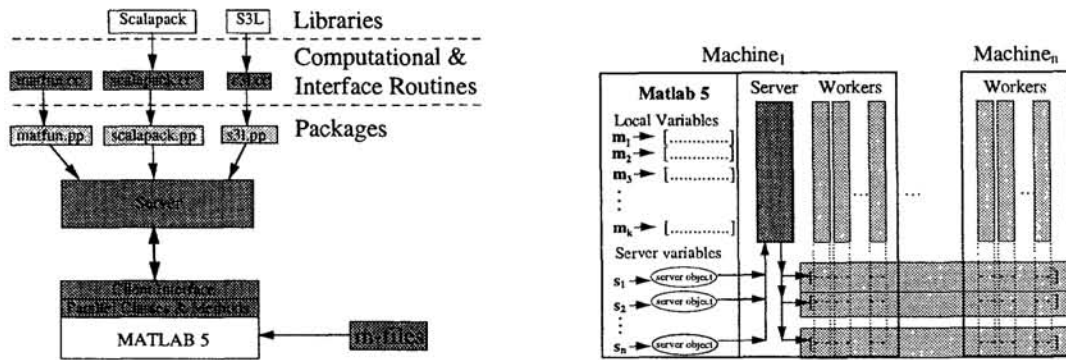

Figure 1: Use of the PPServer by Matlab is almost completely transparent. PPServer variables are tied to the PPServer itself while Matlab maintains *handles* to the data. Using Matlab's object system, functions using PPServer variables invoke PPServer commands implicitly.

their reach, we discover interesting analyses of both data and algorithms.

## 2   The Parallel Problems Server

The Parallel Problems Server (PPServer) is the foundation of this work. The PPServer is a realization of a novel client-server model for computation on very large matrices. It is compatible with any Unix-like platform supporting the Message Passing Interface (MPI) library (Gropp, Lusk and Skjellum, 1994). MPI is the standard for multi-processor communication and is the most portable way for writing parallel code.

The PPServer implements functions for creating and removing distributed matrices, loading and storing them from/to disk using a portable format, and performing elementary matrix operations. Matrices are two-dimensional single or double precision arrays created on the PPServer itself (functions are provided for transferring matrix sections to and from a client). The PPServer supports both dense and sparse matrices.

The PPServer communicates with clients using a simple request-response protocol. A client requests an action by issuing a command with the appropriate arguments, the server executes that command, and then notifies the client that the command is complete.

The PPServer is directly extensible via compiled libraries called *packages*. The PPServer implements a robust protocol for communicating with packages. Clients (and other packages) can load and remove packages on-the-fly, as well as execute commands within packages.

Package programmers have direct access to information about the PPServer and its matrices. Each package represents its own namespace, defining a set of visible function names. This supports data encapsulation and allows users to hide a subset of functions in one package by loading another that defines the same function names. Finally, packages support common parallel idioms (*eg* applying a function to every element of a matrix), making it easier to add common functionality.

All but a few PPServer commands are implemented in packages, including basic matrix operations. Many highly-optimized public libraries have been realized as packages using appropriate wrapper functions. These packages include ScaLAPACK (Blackford et al., 1997), S3L (Sun's optimized version of ScaLAPACK), PARPACK (Maschhoff and Sorensen, 1996), and PETSc (PETSc, ).

```
1 function H=hilb(n)
2   J = 1:n;
3   J = J(ones(n,1),:);
4   I = J';
5   E = ones(n,n);
6   H = E./(I+J-1);
```

Figure 2: Matlab code for producing Hilbert matrices. When n is influenced by P, each of the constructors creates a PPServer object instead of a Matlab object.

## 3   MATLAB*P

By directly using the PPServer's client communication interface, it is possible for other applications to use the PPServer's functionality. We have implemented a client interface for Matlab, called MATLAB*P. MATLAB*P is a collection of Matlab 5 objects, Matlab m-files (Matlab's scripting language) and Matlab MEX programs (Matlab's external language API) that allows for the transparent integration of Matlab as a front end for the Parallel Problems Server.

The choice of Matlab was influenced by several factors. It is the *de facto* standard for scientific computing, enjoying wide use in industry and academia. In the machine learning community, for example, algorithms are often written as Matlab scripts and made freely available. In the scientific computing community, algorithms are often first prototyped in Matlab before being optimized for languages such as Fortran.

We endeavor to make interaction with the PPServer as transparent as possible for the user. In principle, a typical Matlab user should never have to make explicit calls to the PPServer. Further, current Matlab programs should not have to be rewritten to take advantage of the PPServer.

Space does not permit a complete discussion of MATLAB*P (we refer the reader to (Husbands and Isbell, 1999)); however, we will briefly discuss how to use prewritten Matlab scripts without modification. This is accomplished through the simple but innovative P notation.

We use Matlab 5's object oriented features to create PPServer objects automatically. P is a special object we introduce in Matlab that acts just like the integer 1. A user typing a=ones(1000*P,1000) or b=rand(1000,1000*P) obtains two 1000-by-1000 matrices distributed in parallel. The reader can guess the use of P here: it indicates that a is distributed by rows and b by columns.

To a user, a and b are matrices, but within Matlab, they are handles to special distributed types that exist on the PPServer. Any further references to these variables (e.g. via such commands as eig, svd, inv, *, +, -) are recognized as a call to the PPServer rather than as a traditional Matlab command.

Figure 2 shows the code for Matlab's built in function hilb. The call hilb(n) produces the $n \times n$ Hilbert matrix ($H_{ij} = \frac{1}{i+j-1}$). When n is influenced by P, a parallel array results:

- J=1:n in line 2 creates the PPServer vector $1, 2, \cdots, n$ and places a handle to it in J. Note that this behavior does not interfere with the semantics of for loops (for i=1:n) as Matlab assigns to i the value of each column of 1:n: the numbers $1, 2, \ldots, n$.
- ones(n,1) in line 3 produces a PPServer matrix.
- Emulation of Matlab's indexing functions results in the correct execution of line 3.

- Overloading of ' (the transpose operator) executes line 4 on the PPServer.
- In line 5, E is generated on the PPServer because of the overloading of ones.
- Overloading elementary matrix operations makes H a PPServer matrix (line 6).

The Parallel Problems Server and MATLAB*P have been tested extensively on a variety of platforms. They currently run on Cray supercomputers[1], clusters of symmetric multiprocessors from Sun Microsystems and DEC as well as on clusters of networked Intel PCs. The PPServer has also been tested with other clients, including Common LISP.

Although computational performance varies depending upon the platform, it is clear that the system provides distinct computational advantages. Communication overhead (in our experiments, roughly two milliseconds per PPServer command) is negligible compared to the computational and space advantage afforded by transparent access to highly-optimized linear algebra algorithms.

# 4   Applications in Text Retrieval

In this section we demonstrate the efficacy of the PPServer on real-world machine learning problems. In particular we explore the use of the PPServer and MATLAB*P in the text retrieval domain.

The task in text retrieval is to find the subset of a collection of documents relevant to a user's information request. Standard approaches are based on the Vector Space Model (VSM). A document is a vector where each dimension is a count of the occurrence of a different word. A collection of documents is a matrix, $D$, where each column is a document vector $d_i$. The similarity between two documents is their inner product, $d_i^T d_j$. Queries are just like documents, so the relevance of documents to a query, $q$, is $D^T q$.

Typical small collections contain a thousand vectors in a ten thousand dimensional space, while large collections may contain 500,000 vectors residing in hundreds of thousands of dimensions. Clearly, well-understood standard machine learning techniques may exhibit unpredictable behavior under such circumstances, or simply may not scale at all.

Classically, ML-like approaches try to construct a set of linear operators which extract the underlying "topic" structure of documents. Documents and queries are projected into that new (usually smaller) space before being compared using the inner product.

The large matrix support in MATLAB*P enables us to use matrix decomposition techniques for extracting linear operators easily. We have explored several different algorithms(Isbell and Viola, 1998). Below, we discuss two standard algorithms to demonstrate how the PPServer allows us to perform interesting analysis on large datasets.

## 4.1   Latent Semantic Indexing

Latent Semantic Indexing (LSI) (Deerwester et al., 1990) constructs a smaller document matrix by using the Singular Value Decomposition (SVD): $D = USV^T$. $U$ contains the eigenvectors of the co-occurrence matrix while the diagonal elements of $S$ (referred to as *singular values*) contain the square roots of their corresponding eigenvalues. The eigenvectors with the largest eigenvalues capture the axes of largest variation in the data.

LSI projects documents onto the $k$-dimensional subspace spanned by the first $k$ columns of $U$ (denoted $U_k$) so that the documents are now: $V_k^T = S_k^{-1} U_k$. Queries are similarly projected. Thus, the document-query scores for LSI can be obtained with simple Matlab code:

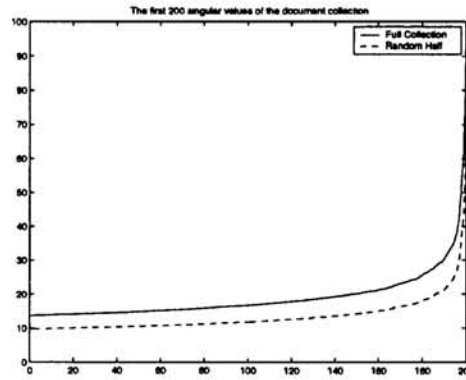

Figure 3: The first 200 singular values of a collection of about 500,000 documents and 200,000 terms, and singular values for half of that collection. Computation for on the full collection took only 62 minutes using 32 processors on a Cray T3E.

```
D=dsparse('term-doc');      %D SPARSE reads a sparse matrix
Q=dsparse('queries');
[U,S,V]=svds(D,k);          % compute the k-SVD of D
sc=getlsiscores(U,S,V,Q);   % computes v*(1/s)*u'*q
```

The scores that are returned can then be combined with relevance judgements to obtain precision/recall curves that are displayed in Matlab:

```
r=dsparse('judgements');
[pr,re]=precisionrecall(sc,r);
plot(re('@'),pr('@'));
```

In addition to evaluating the performance of various techniques, we can also explore characteristics of the data itself. For example, many implementations of LSI on large collections use only a subset of the documents for computational reasons. This leads one to question how the SVD is affected. Figure 3 shows the first singular values for one large collection as well as for a random half of that collection. It shows that the shape of the curves are remarkably similar (as they are for the *other* half). This suggests that we can derive a projection matrix from just half of the collection. An evaluation of this technique can easily be performed using our system. Premlinary experiments show nearly identical retrieval performance.

### 4.2 What are the Independent Components of Documents?

Independent components analysis (ICA)(Bell and Sejnowski, 1995) also recovers linear projections from data. Unlike LSI, which finds principal components, ICA finds axes that are statistically independent. ICA's biggest success is probably its application to the *blind source separation* or *cocktail party* problem. In this problem, one observes the output of a number of microphones. Each microphone is assumed to be recording a linear mixture of a number of unknown sources. The task is to recover the original sources.

There is a natural embedding of text retrieval within this framework. The words that are observed are like microphone signals, and underlying "topics" are the source signals that give rise to them.

Figure 4 shows a typical distribution of words projected along axes found by ICA.[2] Most words have a value close to zero. The histogram shows only the words large positive or

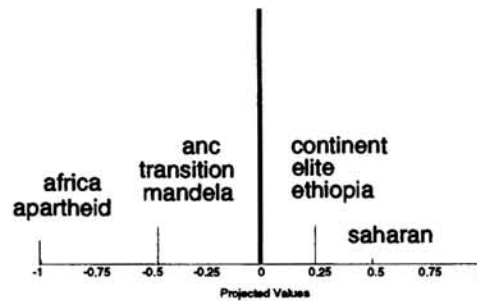

Figure 4: Distribution of words with large magnitude an ICA axis from White House text.

negative values. One group of words is made up of highly-related terms; namely, "africa," "apartheid," and "mandela." The other group of words are not directly related, but each co-occurs with different individual words in the first group. For example, "saharan" and "africa" occur together many times, but not in the context of apartheid and South Africa; rather, in documents concerning US policy toward Africa in general. As it so happens, "saharan" acts as a discriminating word for these subtopics.

As observed in (Isbell and Viola, 1998), it appears that ICA is finding a set of words, $S$, that selects for related documents, $H$, along with another set of words, $T$, whose elements do not select for $H$, but co-occur with elements of $S$. Intuitively, $S$ selects for documents in a general subject area, and $T$ removes a specific subset of those documents, leaving a small set of highly related documents. This suggests a straightforward algorithm to achieve the same goal directly. This local clustering approach is similar to an unsupervised version of Rocchio with Query Zoning (Singhal, 1997).

Further analysis of ICA on similar collections reveals other interesting behavior on large datasets. For example, it is known that ICA will attempt to find an unmixing matrix that is full rank. This is in conflict with the notion that these collections actually reside in a much smaller subspace. We have found in our experiments with ICA that some axes are highly *kurtotic* while others produce gaussian-like distributions. We conjecture that any axis that results in a gaussian-like distribution will be split arbitrarily among all "empty" axes. For all intents and purposes, these axes are uninformative. This provides an automatic noise-reduction technique for ICA when applied to large datasets.

For the purposes of comparison, Figure 5 illustrates the performance of several algorithms (including ICA and various clustering techniques) on articles from the Wall Street Journal.[3]

## 5    Discussion

We have shown that MATLAB*P enables portable, high-performance interactive supercomputing using the Parallel Problems Server, a powerful mechanism for writing and accessing optimized algorithms. Further, the client communication protocol makes it possible to implement transparent integration with sufficiently powerful clients, such as Matlab 5.

With such a tool, researchers can now use Matlab as something more than just a way for prototyping algorithms and working on small problems. MATLAB*P makes it possible to interactively operate on and visualize large data sets. We have demonstrated this last claim by using the PPServer system to apply ML techniques to large datasets, allowing for analyses of both data and algorithms. MATLAB*P has also been used to implement versions of Diverse Density(Maron, 1998), MIMIC(DeBonet, Isbell and Viola, 1996), and gradient descent.

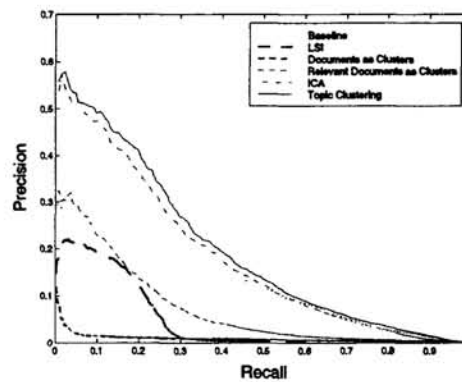

Figure 5: A comparison of different algorithms on the Wall Street Journal

## Footnotes

[1]Although there is no Matlab for the Cray, we are still able to use it to "execute" Matlab code in parallel.

[2] These results are from a collection containing transcripts of White House press releases from 1993. There are 1585 documents and 18,675 distinct words.

[3]The WSJ collection contains 42,652 documents and 89,757 words

## References

Bell, A. and Sejnowski, T. (1995). An information-maximizaton approach to blind source separation and blind deconvolution. *Neural Computation*, 7:1129–1159.

Blackford, L. S., Choi, J., Cleary, A., D'Azevedo, E., Demmel, J., Dhilon, I., Dongarra, J., Hammarling, S., Henry, G., Petitet, A., Stanley, K., Walker, D., and Whaley, R. (1997). ScaLAPACK Users' Guide. http://www.netlib.org/scalapack/slug/scalapack_slug.html.

DeBonet, J., Isbell, C., and Viola, P. (1996). Mimic: Finding optima by estimating probability densities. In *Advances in Neural Information Processing Systems*.

Deerwester, S., Dumais, S. T., Landauer, T. K., Furnas, G. W., and Harshman, R. A. (1990). Indexing by latent semantic analysis. *Journal of the Society for Information Science*, 41(6):391–407.

Frakes, W. B. and Baeza-Yates, R., editors (1992). *Information Retrieval: Data Structures and Algorithms*. Prentice-Hall.

Gropp, W., Lusk, E., and Skjellum, A. (1994). *Using MPI: Portable Parallel Programming with the Message-Passing Interface*. The MIT Press.

Husbands, P. and Isbell, C. (1999). MITMatlab: A tool for interactive supercomputing. In *Proceedings of the Ninth SIAM Conference on Parallel Processing for Scientific Computing*.

Isbell, C. and Viola, P. (1998). Restructuring sparse high dimensional data for effective retrieval. In *Advances in Neural Information Processing Systems*.

Kwok, K. L. (1996). A new method of weighting query terms for ad-hoc retrieval. In *Proceedings of the 19th ACM/SIGIR Conference*, pages 187–195.

Maron, O. (1998). A framework for multiple-instance learning. In *Advances in Neural Information Processing Systems*.

Maschhoff, K. J. and Sorensen, D. C. (1996). A Portable Implementation of ARPACK for Distributed Memory Parallel Computers. In *Preliminary Proceedings of the Copper Mountain Conference on Iterative Methods*.

O'Brien, G. W. (1994). Information management tools for updating an svd-encoded indexing scheme. Technical Report UT-CS-94-259, University of Tennessee.

PETSc. The Portable, Extensible Toolkit for Scientific Computation. http://www.mcs.anl.gov/home/group/petsc.html.

PPServer. The Parallel Problems Server Web Page. http://www.ai.mit.edu/projects/ppserver.

Sahami, M., Hearst, M., and Saund, E. (1996). Applying the multiple cause mixture model to text categorization. In *Proceedings of the 13th International Machine Learning Conference*.

Singhal, A. (1997). Learning routing queries in a query zone. In *Proceedings of the 20th International Conference on Research and Development in Information Retrieval*.
